# Integrated Modeling and Control Based on Reinforcement Learning and Dynamic Programming

**Richard S. Sutton**
GTE Laboratories Incorporated
Waltham, MA 02254

## Abstract

This is a summary of results with Dyna, a class of architectures for intelligent systems based on approximating dynamic programming methods. Dyna architectures integrate trial-and-error (reinforcement) learning and execution-time planning into a single process operating alternately on the world and on a learned forward model of the world. We describe and show results for two Dyna architectures, Dyna-AHC and Dyna-Q. Using a navigation task, results are shown for a simple Dyna-AHC system which simultaneously learns by trial and error, learns a world model, and plans optimal routes using the evolving world model. We show that Dyna-Q architectures (based on Watkins's Q-learning) are easy to adapt for use in changing environments.

## 1 Introduction to Dyna

Dyna architectures (Sutton, 1990) use learning algorithms to approximate the conventional optimal control technique known as *dynamic programming (DP)* (Bellman, 1957; Bertsekas, 1987). DP itself is not a learning method, but rather a computational method for determining optimal behavior given a complete model of the task to be solved. It is very similar to state-space search, but differs in that it is more incremental and never considers actual action *sequences* explicitly, only single actions at a time. This makes DP more amenable to incremental planning at execution time, and also makes it more suitable for stochastic or incompletely modeled environments, as it need not consider the extremely large number of sequences possible in an uncertain environment. Learned world models are likely to be stochastic and uncertain, making DP approaches particularly promising for

learning systems. Dyna architectures are those that learn a world model online while using approximations to DP to learn and plan optimal behavior.

The theory of Dyna is based on the theory of DP and on DP's relationship to reinforcement learning (Watkins, 1989; Barto, Sutton & Watkins, 1989, 1990), to temporal-difference learning (Sutton, 1988), and to AI methods for planning and search (Korf, 1990). Werbos (1987) has previously argued for the general idea of building AI systems that approximate dynamic programming, and Whitehead & Ballard (1989) and others (Sutton & Barto, 1981; Sutton & Pinette, 1985; Rumelhart et al., 1986; Lin, 1991; Riolo, 1991) have presented results for the specific idea of augmenting a reinforcement learning system with a world model used for planning.

## 2  Dyna-AHC: Dyna by Approximating Policy Iteration

The *Dyna-AHC* architecture is based on approximating a DP method known as policy iteration (see Bertsekas, 1987). It consists of four components interacting as shown in Figure 1. The *policy* is simply the function formed by the current set of reactions; it receives as input a description of the current state of the world and produces as output an action to be sent to the world. The *world* represents the task to be solved; prototypically it is the robot's external environment. The world receives actions from the policy and produces a next state output and a reward output. The overall task is defined as maximizing the long-term average reward per time step. The architecture also includes an explicit *world model*. The world model is intended to mimic the one-step input-output behavior of the real world. Finally, the Dyna-AHC architecture includes an *evaluation function* that rapidly maps states to values, much as the policy rapidly maps states to actions. The evaluation function, the policy, and the world model are each updated by separate learning processes.

The policy is continually modified by an integrated planning/learning process. The policy is, in a sense, a *plan*, but one that is completely conditioned by current input. The planning process is incremental and can be interrupted and resumed at any time. It consists of a series of shallow seaches, each typically of one ply, and yet ultimately produces the same result as an arbitrarily deep conventional search. I call this *relaxation planning*.

Relaxation planning is based on continually adjusting the evaluation function in such a way that credit is propagated to the appropriate steps within action sequences. Generally speaking, the evaluation $e(x)$ of a state $x$ should be equal to the best of the states $y$ that can be reached from it in one action, taking into consideration the reward (or cost) $r$ for that one transition:

$$e(x) \quad \text{``} = \text{''} \quad \max_{a \in Actions} E\{r + e(y) \mid x, a\}, \tag{1}$$

where $E\{\cdot \mid \cdot\}$ denotes a conditional expected value and the equal sign is quoted to indicate that this is a condition that we would like to hold, not one that necessarily does hold. If we have a complete model of the world, then the right-hand side can be computed by looking ahead one action. Thus we can generate any number of training examples for the process that learns the evaluation function: for any $x$,

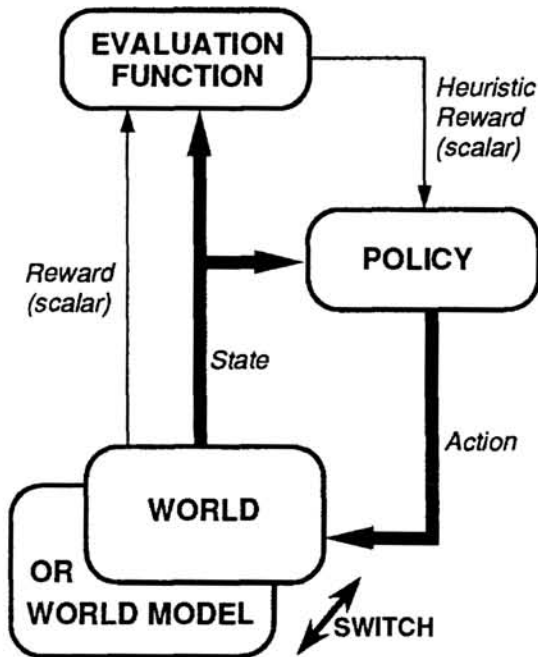

Figure 1. Overview of Dyna-AHC

1. Decide if this will be a real experience or a hypothetical one.
2. Pick a state $x$. If this is a real experience, use the current state.
3. Choose an action: $a \leftarrow Policy(x)$
4. Do action $a$; obtain next state $y$ and reward $r$ from world or world model.
5. If this is a real experience, update world model from $x$, $a$, $y$ and $r$.
6. Update evaluation function so that $e(x)$ is more like $r + \gamma e(y)$; this is temporal-difference learning.
7. Update policy—strengthen or weaken the tendency to perform action $a$ in state $x$ according to the error in the evaluation function: $r + \gamma e(y) - e(x)$.
8. Go to Step 1.

Figure 2. Inner Loop of Dyna-AHC. These steps are repeatedly continually, sometimes with real experiences, sometimes with hypothetical ones.

the right-hand side of (1) is the desired output. If the learning process converges such that (1) holds in all states, then the optimal policy is given by choosing the action in each state $x$ that achieves the maximum on the right-hand side. There is an extensive theoretical basis from dynamic programming for algorithms of this type for the special case in which the evaluation function is tabular, with enumerable states and actions. For example, this theory guarantees convergence to a unique evaluation function satisfying (1) and that the corresponding policy is optimal (Bertsekas, 1987).

The evaluation function and policy need not be tables, but can be more compact function approximators such as connectionist networks, decision trees, $k$-$d$ trees, or symbolic rules. Although the existing theory does not apply to these machine learning algorithms directly, it does provide a theoretical foundation for exploring their use in this way.

The above discussion gives the general idea of relaxation planning, but not the exact form used in policy iteration and Dyna-AHC, in which the policy is adapted simultaneously with the evaluation function. The evaluations in this case are not supposed to reflect the value of states given optimal behavior, but rather their value given current behavior (the current policy). As the current policy gradually approaches optimality, the evaluation function also approaches the optimal evaluation function. In addition, Dyna-AHC is a *Monte Carlo* or *stochastic approximation* variant of policy iteration, in which the world model is only sampled, not examined directly. Since the real world can also be sampled, by actually taking actions and observing the result, the world can be used in place of the world model in these methods. In this case, the result is not relaxation planning, but a trial-and-error learning process much like reinforcement learning (see Barto, Sutton & Watkins,

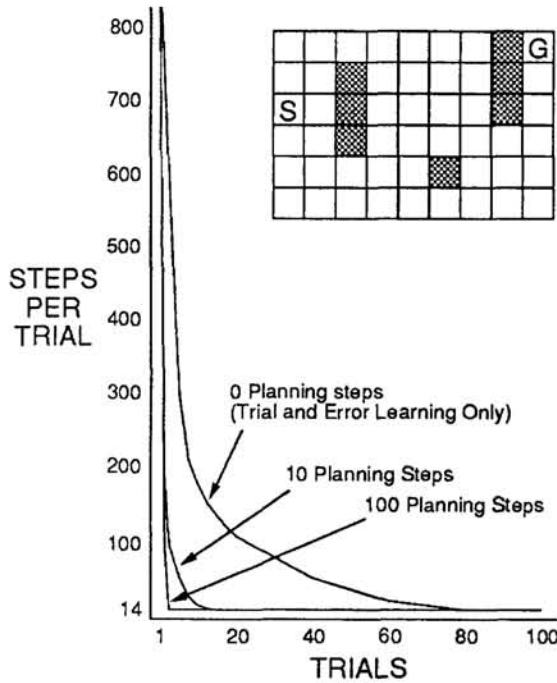

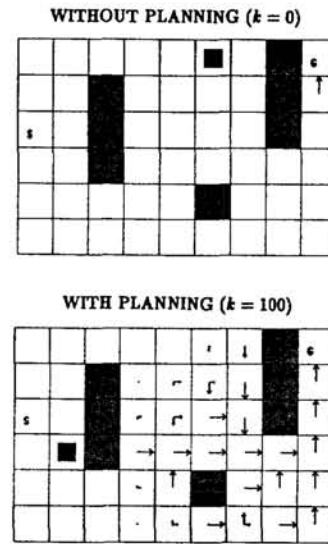

WITHOUT PLANNING (k = 0)

WITH PLANNING (k = 100)

Figure 3.  Learning Curves of Dyna-AHC Systems on a Navigation Task

Figure 4.  Policies Found by Planning and Non-Planning Dyna-AHC Systems by the Middle of the Second Trial. The black square is the current location of the system. The arrows indicate action probabilities (excess over smallest) for each direction of movement.

1989, 1990). In Dyna-AHC, both of these are done at once. The same algorithm is applied both to real experience (resulting in learning) and to hypothetical experience generated by the world model (resulting in relaxation planning). The results in both cases are accumulated in the policy and the evaluation function.

There is insufficient room here to fully justify the algorithm used in Dyna-AHC, but it is quite simple and is given in outline form in Figure 2.

## 3  A Navigation Task

As an illustration of the Dyna-AHC architecture, consider the task of navigating the maze shown in the upper right of Figure 3. The maze is a 6 by 9 grid of possible locations or states, one of which is marked as the starting state, "S", and one of which is marked as the goal state, "G". The shaded states act as barriers and cannot be entered. All the other states are distinct and completely distinguishable. From each there are four possible actions: UP, DOWN, RIGHT, and LEFT, which change the state accordingly, except where such a movement would take the take the system into a barrier or outside the maze, in which case the location is not changed. Reward is zero for all transitions except for those into the goal state, for which it is +1. Upon entering the goal state, the system is instantly transported back to the start state to begin the next trial. None of this structure and dynamics is known to the Dyna-AHC system a priori.

In this instance of the Dyna-AHC architecture, real and hypothetical experiences

were used alternately (Step 1). For each single experience with the real world, $k$ hypothetical experiences were generated with the model. Figure 3 shows learning curves for $k = 0$, $k = 10$, and $k = 100$, each an average over 100 runs. The $k = 0$ case involves no planning; this is a pure trial-and-error learning system entirely analogous to those used in reinforcement learning systems based on the adaptive heuristic critic (AHC) (Sutton, 1984; Barto, Sutton & Anderson, 1983). Although the length of path taken from start to goal falls dramatically for this case, it falls much *more* rapidly for the cases including hypothetical experiences, showing the benefit of relaxation planning using the learned world model. For $k = 100$, the optimal path was generally found and followed by the fourth trip from start to goal; this is very rapid learning.

Figure 4 shows why a Dyna-AHC system that includes planning solves this problem so much faster than one that does not. Shown are the policies found by the $k = 0$ and $k = 100$ Dyna-AHC systems half-way through the second trial. Without planning ($k = 0$), each trial adds only one additional step to the policy, and so only one step (the last) has been learned so far. With planning, the first trial also learned only one step, but here during the second trial an extensive policy has been developed that by the trial's end will reach almost back to the start state.

## 4   Dyna-Q: Dyna by Q-learning

The Dyna-AHC architecture is in essence the reinforcement learning architecture based on the adaptive heuristic critic (AHC) that my colleagues and I developed (Sutton, 1984; Barto, Sutton & Anderson, 1983) *plus* the idea of using a learned world model to generate hypothetical experience and to plan. Watkins (1989) subsequently developed the relationships between the reinforcement-learning architecture and dynamic programming (see also Barto, Sutton & Watkins, 1989, 1990) and, moreover, proposed a slightly different kind of reinforcement learning called *Q-learning*. The *Dyna-Q* architecture is the combination of this new kind of learning with the Dyna idea of using a learned world model to generate hypothetical experience and achieve planning.

Whereas the AHC reinforcement learning architecture maintains two fundamental memory structures, the evaluation function and the policy, Q-learning maintains only one. That one is a cross between an evaluation function and a policy. For each pair of state $x$ and action $a$, Q-learning maintains an estimate $Q_{xa}$ of the value of taking $a$ in $x$. The value of a *state* can then be defined as the value of the state's best state-action pair: $e(x) \stackrel{\text{def}}{=} \max_a Q_{xa}$. In general, the Q-value for a state $x$ and an action $a$ should equal the expected value of the immediate reward $r$ plus the discounted value of the next state $y$:

$$Q_{xa} \ \text{``}=\text{''} \ E\left\{r + \gamma e(y) \mid x, a\right\}. \tag{3}$$

To achieve this goal, the updating steps (Steps 6 and 7 of Figure 2) are implemented by

$$Q_{xa} \leftarrow Q_{xa} + \beta\big(r + \gamma e(y) - Q_{xa}\big). \tag{4}$$

This is the only update rule in Q-learning. We note that it is very similar though not identical to Holland's bucket brigade and to Sutton's (1988) temporal-difference learning.

The simplest way of determining the policy on real experiences is to deterministically select the action that currently looks best—the action with the maximal Q-value. However, as we show below, this approach alone suffers from inadequate exploration. To deal with this problem, a new memory structure was added that keeps track of the degree of uncertainty about each component of the model. For each state $x$ and action $a$, a record is kept of the number of time steps $n_{xa}$ that have elapsed since $a$ was tried in $x$ in a real experience. An exploration bonus of $\epsilon\sqrt{n_{xa}}$ is used to make actions that have not been tried in a long time (and that therefore have uncertain consequences) appear more attractive by replacing (4) with:

$$Q_{xa} \leftarrow Q_{xa} + \beta\left(r + \epsilon\sqrt{n_{xa}} + \gamma e(y) - Q_{xa}\right). \tag{5}$$

In addition, the system is permitted to hypothetically experience actions is has never before tried, so that the exploration bonus for trying them can be propagated back by relaxation planning. This was done by starting the system with a non-empty initial model and by selecting actions randomly on hypothetical experiences. In the experiments with Dyna-Q systems reported below, actions that had never been tried were assumed to produce zero reward and leave the state unchanged.

## 5    Changing-World Experiments

Two experiments were performed to test the ability of Dyna systems to adapt to changes in their environments. Three Dyna systems were used: the Dyna-AHC system presented earlier in the paper, a Dyna-Q system including the exploration bonus (5), called the *Dyna-Q+* system, and a Dyna-Q system without the exploration bonus (4), called the *Dyna-Q-* system. All systems used $k = 10$.

The *blocking experiment* used the two mazes shown in the upper portion of Figure 5. Initially a short path from start to goal was available (first maze). After 1000 time steps, by which time the short path was usually well learned, that path was blocked and a longer path was opened (second maze). Performance under the new condition was measured for 2000 time steps. Average results over 50 runs are shown in Figure 5 for the three Dyna systems. The graph shows a *cumulative* record of the number of rewards received by the system up to each moment in time. In the first 1000 trials, all three Dyna systems found a short route to the goal, though the Dyna-Q+ system did so significantly faster than the other two. After the short path was blocked at 1000 steps, the graph for the Dyna-AHC system remains almost flat, indicating that it was unable to obtain further rewards. The Dyna-Q systems, on the other hand, clearly solved the blocking problem, reliably finding the alternate path after about 800 time steps.

The *shortcut experiment* began with only a long path available (first maze of Figure 6). After 3000 times steps all three Dyna systems had learned the long path, and then a shortcut was opened without interferring with the long path (second maze of Figure 6). The lower part of Figure 6 shows the results. The increase in the slope of the curve for the Dyna-Q+ system, while the others remain constant, indicates that it alone was able to find the shortcut. The Dyna-Q+ system also learned the original long route faster than the Dyna-Q- system, which in turn learned it faster than the Dyna-AHC system. However, the ability of the Dyna-Q+ system to find shortcuts does not come totally for free. Continually re-exploring the world

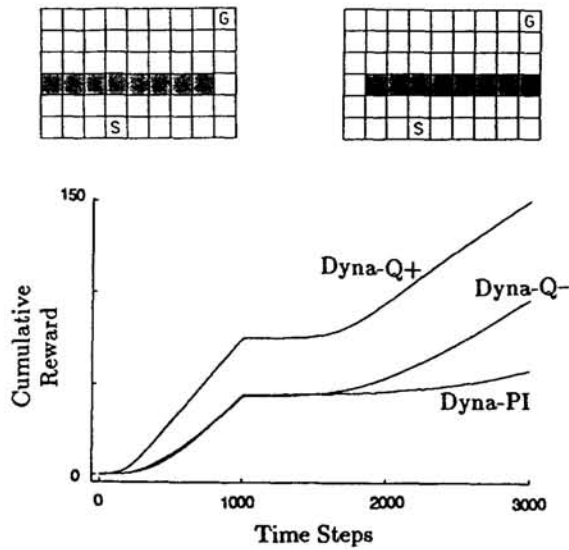

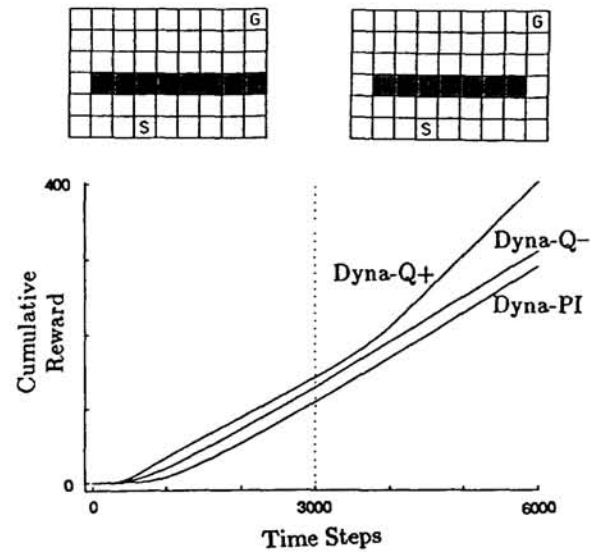

Figure 5. Performance on the Blocking Task (Slope is the *Rate* of Reward)

Figure 6. Performance on the Shortcut Task (Slope is the *Rate* of Reward)

means occasionally making suboptimal actions. If one looks closely at Figure 6, one can see that the Dyna-Q+ system actually acheives a slightly lower *rate* of reinforcement during the first 3000 steps. In a static environment, Dyna-Q+ will eventually perform worse than Dyna-Q-, whereas, in a changing environment, it will be far superior, as here. One possibility is to use a meta-level learning process to adjust the exploration parameter $\epsilon$ to match the degree of variability of the environment.

## 6   Limitations and Conclusions

The results presented here are clearly limited in many ways. The state and action spaces are small and denumerable, permitting tables to be used for all learning processes and making it feasible for the entire state space to be explicitly explored. In addition, these results have assumed knowledge of the world state, have used a trivial form of search control (random exploration), and have used terminal goal states. These are significant limitations of the results, but not of the Dyna architecture. There is nothing about the Dyna architecture which prevents it from being applied more generally in each of these ways (e.g., see Lin, 1991; Riolo, 1991; Whitehead & Ballard, in press).

Despite limitations, these results are significant. They show that the use of a forward model can dramatically speed trial-and-error (reinforcement) learning processes even on simple problems. Moreover, they show how planning can be done with the incomplete, changing, and oftimes incorrect world models that are contructed through learning. Finally, they show how the functionality of planning can be obtained in a completely incremental manner, and how a planning process can be freely intermixed with reaction and learning processes. Further results are needed for a thorough comparison of Dyna-AHC and Dyna-Q architectures, but the results presented here suggest that it is easier to adapt Dyna-Q architectures to changing environments.

## Acknowledgements

The author gratefully acknowledges the contributions by Andrew Barto, Chris Watkins, Steve Whitehead, Paul Werbos, Luis Almeida, and Leslie Kaelbling.

## References

Barto, A. G., Sutton R. S., & Anderson, C. W. (1983) *IEEE Trans. SMC-13*, 834–846.

Barto, A. G., Sutton, R. S., & Watkins, C. J. C. H. (1989) In: *Learning and Computational Neuroscience*, M. Gabriel and J.W. Moore (Eds.), MIT Press, 1991.

Barto, A. G., Sutton, R. S., & Watkins, C. J. C. H. (1990) *NIPS 2*, 686–693.

Bellman, R. E. (1957) *Dynamic Programming*, Princeton University Press.

Bertsekas, D. P. (1987) *Dynamic Programming: Deterministic and Stochastic Models*, Prentice-Hall.

Korf, R. E. (1990) *Artificial Intelligence 42*, 189–211.

Lin, Long-Ji. (1991) In: *Proceedings of the International Conference on the Simulation of Adaptive Behavior*, MIT Press.

Riolo, R. (1991) In: *Proceedings of the International Conference on the Simulation of Adaptive Behavior*, MIT Press.

Rumelhart, D. E., Smolensky, P., McClelland, J. L., & Hinton, G. E. (1986) In: *Parallel Distributed Processing: Explorations in the Microstructure of Cognition, Volume II*, by J. L. McClelland, D. E. Rumelhart, and the PDP research group, 7–57. MIT Press.

Sutton, R. S. (1984) *Temporal Credit Assignment in Reinforcement Learning*. PhD thesis, COINS Dept., Univ, of Mass.

Sutton, R.S. (1988) *Machine Learning 3*, 9–44.

Sutton, R.S. (1990) In: Proceedings of the Seventh International Conference on Machine Learning, 216–224, Morgan Kaufmann.

Sutton, R.S., Barto, A.G. (1981) *Cognition and Brain Theory 4*, 217–246.

Sutton, R.S., Pinette, B. (1985) In: *Proceedings of the Seventh Annual Conf. of the Cognitive Science Society*, 54–64, Lawrence Erlbaum.

Watkins, C. J. C. H. (1989) *Learning with Delayed Rewards*. PhD thesis, Cambridge University Psychology Department.

Werbos, P. J. (1987) *IEEE Trans. SMC-17*, 7–20.

Whitehead, S. D., Ballard, D.H. (1989) In: *Proceedings of the Sixth International Workshop on Machine Learning*, 354–357, Morgan Kaufmann.

Whitehead, S. D., Ballard, D.H. (in press) *Machine Learning*.